# A New Approximate Maximal Margin Classification Algorithm

**Claudio Gentile**
DSI, Universita' di Milano,
Via Comelico 39,
20135 Milano, Italy
gentile@dsi.unimi.it

## Abstract

A new incremental learning algorithm is described which approximates the maximal margin hyperplane w.r.t. norm $p \geq 2$ for a set of linearly separable data. Our algorithm, called ALMA$_p$ (Approximate Large Margin algorithm w.r.t. norm $p$), takes $O\left(\frac{(p-1)\,X^2}{\alpha^2\,\gamma^2}\right)$ corrections to separate the data with $p$-norm margin larger than $(1-\alpha)\,\gamma$, where $\gamma$ is the $p$-norm margin of the data and $X$ is a bound on the $p$-norm of the instances. ALMA$_p$ avoids quadratic (or higher-order) programming methods. It is very easy to implement and is as fast as on-line algorithms, such as Rosenblatt's perceptron. We report on some experiments comparing ALMA$_p$ to two incremental algorithms: Perceptron and Li and Long's ROMMA. Our algorithm seems to perform quite better than both. The accuracy levels achieved by ALMA$_p$ are slightly inferior to those obtained by Support vector Machines (SVMs). On the other hand, ALMA$_p$ is quite faster and easier to implement than standard SVMs training algorithms.

## 1 Introduction

A great deal of effort has been devoted in recent years to the study of maximal margin classifiers. This interest is largely due to their remarkable generalization ability. In this paper we focus on special maximal margin classifiers, i.e., on maximal margin hyperplanes. Briefly, given a set linearly separable data, the maximal margin hyperplane classifies all the data correctly and maximizes the minimal distance between the data and the hyperplane. If euclidean norm is used to measure the distance then computing the maximal margin hyperplane corresponds to the, by now classical, Support Vector Machines (SVMs) training problem [3]. This task is naturally formulated as a quadratic programming problem. If an arbitrary norm $p$ is used then such a task turns to a more general mathematical programming problem (see, e.g., [15, 16]) to be solved by general purpose (and computationally intensive) optimization methods. This more general task arises in feature selection problems when the target to be learned is sparse.

A major theme of this paper is to devise simple and efficient algorithms to solve the maximal margin hyperplane problem. The paper has two main contributions. The first contribution is a new efficient algorithm which *approximates* the maximal margin hyperplane w.r.t.

norm $p$ to any given accuracy. We call this algorithm ALMA$_p$ (Approximate Large Margin algorithm w.r.t. norm $p$). ALMA$_p$ is naturally viewed as an *on-line* algorithm, i.e., as an algorithm which processes the examples one at a time. A distinguishing feature of ALMA$_p$ is that its relevant parameters (such as the learning rate) are dynamically adjusted over time. In this sense, ALMA$_p$ is a refinement of the on-line algorithms recently introduced in [2]. Moreover, ALMA$_2$ (i.e., ALMA$_p$ with $p = 2$) is a perceptron-like algorithm; the operations it performs can be expressed as dot products, so that we can replace them by kernel functions evaluations. ALMA$_2$ approximately solves the SVMs training problem, avoiding quadratic programming. As far as theoretical performance is concerned, ALMA$_2$ achieves essentially the same bound on the number of corrections as the one obtained by a version of Li and Long's ROMMA algorithm [12], though the two algorithms are different.[1] In the case that $p$ is logarithmic in the dimension of the instance space (as in [6]) ALMA$_p$ yields results which are similar to those obtained by estimators based on linear programming (see [1, Chapter 14]).

The second contribution of this paper is an experimental investigation of ALMA$_2$ on the problem of handwritten digit recognition. For the sake of comparison, we followed the experimental setting described in [3, 4, 12]. We ran ALMA$_2$ with polynomial kernels, using both the last and the voted hypotheses (as in [4]), and we compared our results to those described in [3, 4, 12]. We found that voted ALMA$_2$ generalizes quite better than both ROMMA and the voted Perceptron algorithm, but slightly worse than standard SVMs. On the other hand ALMA$_2$ is much faster and easier to implement than standard SVMs training algorithms.

For related work on SVMs (with $p = 2$), see Friess et al. [5], Platt [17] and references therein.

The next section defines our major notation and recalls some basic preliminaries. In Section 3 we describe ALMA$_p$ and claim its theoretical properties. Section 4 describes our experimental comparison. Concluding remarks are given in the last section.

## 2 Preliminaries and notation

An *example* is a pair $(x, y)$, where $x$ is an *instance* belonging to $\mathcal{R}^n$ and $y \in \{-1, +1\}$ is the binary *label* associated with $x$. A weight vector $w = (w_1, ..., w_n) \in \mathcal{R}^n$ represents an $n$-dimensional hyperplane passing through the origin. We associate with $w$ a linear threshold classifier with threshold zero: $w : x \to \text{sign}(w \cdot x) = 1$ if $w \cdot x \geq 0$ and $= -1$ otherwise. When $p \geq 1$ we denote by $||w||_p$ the $p$-norm of $w$, i.e., $||w||_p = (\sum_{i=1}^n |w_i|^p)^{1/p}$ (also, $||w||_\infty = \lim_{p \to \infty} (\sum_{i=1}^n |w_i|^p)^{1/p} = \max_i |w_i|$). We say that $q$ is *dual* to $p$ if $\frac{1}{p} + \frac{1}{q} = 1$ holds. For instance, the 1-norm is dual to the $\infty$-norm and the 2-norm is self-dual. In this paper we assume that $p$ and $q$ are some pair of dual values, with $p \geq 2$. We use $p$-norms for instances and $q$-norms for weight vectors. The (normalized) $p$-norm margin (or just the margin, if $p$ is clear from the surrounding context) of a hyperplane $w$ with $||w||_q \leq 1$ on example $(x, y)$ is defined as $\frac{y\,w \cdot x}{||x||_p}$. If this margin is positive[2] then $w$ classifies $(x, y)$ correctly. Notice that from Hölder's inequality we have $y\,w \cdot x \leq |w \cdot x| \leq ||w||_q ||x||_p \leq ||x||_p$. Hence $\frac{y\,w \cdot x}{||x||_p} \in [-1, 1]$.

Our goal is to approximate the maximal $p$-norm margin hyperplane for a set of examples (the training set). For this purpose, we use terminology and analytical tools from the on-line

learning literature. We focus on an on-line learning model introduced by Littlestone [14]. An on-line learning algorithm processes the examples one at a time in *trials*. In each trial, the algorithm observes an instance $\boldsymbol{x}$ and is required to predict the label $y$ associated with $\boldsymbol{x}$. We denote the prediction by $\hat{y}$. The prediction $\hat{y}$ combines the current instance $\boldsymbol{x}$ with the current internal state of the algorithm. In our case this state is essentially a weight vector $\boldsymbol{w}$, representing the algorithm's current hypothesis about the maximal margin hyperplane. After the prediction is made, the true value of $y$ is revealed and the algorithm suffers a *loss*, measuring the "distance" between the prediction $\hat{y}$ and the label $y$. Then the algorithm updates its internal state.

In this paper the prediction $\hat{y}$ can be seen as the linear function $\hat{y} = \boldsymbol{w} \cdot \boldsymbol{x}$ and the loss is a margin-based 0-1 Loss: the loss of $\boldsymbol{w}$ on example $(\boldsymbol{x}, y)$ is 1 if $\frac{y \, \boldsymbol{w} \cdot \boldsymbol{x}}{\|\boldsymbol{x}\|_p} \le (1 - \alpha)\, \gamma$ and 0 otherwise, for suitably chosen $\alpha, \gamma \in [0, 1]$. Therefore, if $\|\boldsymbol{w}\|_q \le 1$ then the algorithm incurs positive loss if and only if $\boldsymbol{w}$ classifies $(\boldsymbol{x}, y)$ with ($p$-norm) margin not larger than $(1 - \alpha)\, \gamma$. The on-line algorithms are typically *loss driven*, i.e., they do update their internal state only in those trials where they suffer a positive loss. We call a *correction* a trial where this occurs. In the special case when $\alpha = 1$ a correction is a *mistaken* trial and a loss driven algorithm turns to a *mistake driven* [14] algorithm. Throughout the paper we use the subscript $t$ for $\boldsymbol{x}$ and $y$ to denote the instance and the label processed in trial $t$. We use the subscript $k$ for those variables, such as the algorithm's weight vector $\boldsymbol{w}$, which are updated only within a correction. In particular, $\boldsymbol{w}_k$ denotes the algorithm's weight vector after $k-1$ corrections (so that $\boldsymbol{w}_1$ is the initial weight vector). The goal of the on-line algorithm is to bound the cumulative loss (i.e., the total number of corrections or mistakes) it suffers on an arbitrary sequence of examples $S = (\boldsymbol{x}_1, y_1), ..., (\boldsymbol{x}_T, y_T)$. If $S$ is linearly separable with margin $\gamma$ and we pick $\alpha < 1$ then a bounded loss clearly implies convergence in a finite number of steps to (an approximation of) the maximal margin hyperplane for $S$.

## 3 The approximate large margin algorithm $\text{ALMA}_p$

$\text{ALMA}_p$ is a large margin variant of the $p$-norm Perceptron algorithm [3] [8, 6], and is similar in spirit to the variable learning rate algorithms introduced in [2]. We analyze $\text{ALMA}_p$ by giving upper bounds on the number of corrections.

The main theoretical result of this paper is Theorem 1 below. This theorem has two parts. Part 1 bounds the number of corrections in the linearly separable case only. In the special case when $p = 2$ this bound is very similar to the one proven by Li and Long for a version of ROMMA [12]. Part 2 holds for an arbitrary sequence of examples. A bound which is very close to the one proven in [8, 6] for the (constant learning rate) $p$-norm Perceptron algorithm is obtained as a special case. Despite this theoretical similarity, the experiments we report in Section 4 show that using our margin-sensitive variable learning rate algorithm yields a clear increase in performance.

In order to define our algorithm, we need to recall the following mapping $\mathbf{f}$ from [6] (a $p$-indexing for $\mathbf{f}$ is understood): $\mathbf{f} : \mathcal{R}^n \to \mathcal{R}^n$, $\mathbf{f} = (f_1, ..., f_n)$, where

$$f_i(\boldsymbol{w}) = \text{sign}(w_i) \, |w_i|^{q-1} / \|\boldsymbol{w}\|_q^{q-2}, \quad \boldsymbol{w} = (w_1, ..., w_n) \in \mathcal{R}^n.$$

Observe that $p = 2$ yields the identify function. The (unique) inverse $\mathbf{f}^{-1}$ of $\mathbf{f}$ is [6] $\mathbf{f}^{-1} : \mathcal{R}^n \to \mathcal{R}^n$, $\mathbf{f}^{-1} = (f_1^{-1}, ..., f_n^{-1})$, where

$$f_i^{-1}(\boldsymbol{\theta}) = \text{sign}(\theta_i) \, |\theta_i|^{p-1} / \|\boldsymbol{\theta}\|_p^{p-2}, \quad \boldsymbol{\theta} = (\theta_1, ..., \theta_n) \in \mathcal{R}^n,$$

namely, $\mathbf{f}^{-1}$ is obtained from $\mathbf{f}$ by replacing $q$ with $p$.

**Algorithm** $\text{ALMA}_p(\alpha; B, C)$
with $\alpha \in (0,1], B, C > 0$.

**Initialization:** Initial weight vector $\boldsymbol{w}_1 = \boldsymbol{0}$; $k = 1$.
**For** $t = 1, ..., T$ **do**:
Get example $(\boldsymbol{x}_t, y_t) \in \mathcal{R}^n \times \{-1, +1\}$ and update weights as follows:

Set $\gamma_k = B\sqrt{p-1}\,\frac{1}{\sqrt{k}}$;

If $\frac{y_t\,\boldsymbol{w}_k \cdot \boldsymbol{x}_t}{\|\boldsymbol{x}_t\|_p} \le (1-\alpha)\,\gamma_k$ **then** : $\eta_k = \frac{C}{\sqrt{p-1}\,\|\boldsymbol{x}_t\|_p}\,\frac{1}{\sqrt{k}}$,

$$\boldsymbol{w}'_k = \mathbf{f}^{-1}(\mathbf{f}(\boldsymbol{w}_k) + \eta_k\,y_t\,\boldsymbol{x}_t),$$
$$\boldsymbol{w}_{k+1} = \boldsymbol{w}'_k / \max\{1, \|\boldsymbol{w}'_k\|_q\},$$
$$k \leftarrow k + 1.$$

Figure 1: The approximate large margin algorithm $\text{ALMA}_p$.

$\text{ALMA}_p$ is described in Figure 1. The algorithm is parameterized by $\alpha \in (0,1]$, $B > 0$ and $C > 0$. The parameter $\alpha$ measures the degree of approximation to the optimal margin hyperplane, while $B$ and $C$ might be considered as tuning parameters. Their use will be made clear in Theorem 1 below. Let $\mathcal{W} = \{\boldsymbol{w} \in \mathcal{R}^n : \|\boldsymbol{w}\|_q \le 1\}$. $\text{ALMA}_p$ maintains a vector $\boldsymbol{w}_k$ of $n$ weights in $\mathcal{W}$. It starts from $\boldsymbol{w}_1 = \boldsymbol{0}$. At time $t$ the algorithm processes the example $(\boldsymbol{x}_t, y_t)$. If the current weight vector $\boldsymbol{w}_k$ classifies $(\boldsymbol{x}_t, y_t)$ with margin not larger than $(1-\alpha)\,\gamma_k$ then a correction occurs. The update rule[4] has two main steps. The first step gives $\boldsymbol{w}'_k$ through the classical update of a ($p$-norm) perceptron-like algorithm (notice, however, that the learning rate $\eta_k$ scales with $k$, the number of corrections occurred so far). The second step gives $\boldsymbol{w}_{k+1}$ by *projecting* $\boldsymbol{w}'_k$ onto $\mathcal{W}$: $\boldsymbol{w}_{k+1} = \boldsymbol{w}'_k / \|\boldsymbol{w}'_k\|_q$ if $\|\boldsymbol{w}'_k\|_q > 1$ and $\boldsymbol{w}_{k+1} = \boldsymbol{w}'_k$ otherwise. The projection step makes the new weight vector $\boldsymbol{w}_{k+1}$ belong to $\mathcal{W}$.

The following theorem, whose proof is omitted due to space limitations, has two parts. In part 1 we treat the separable case. Here we claim that a special choice of the parameters $B$ and $C$ gives rise to an algorithm which approximates the maximal margin hyperplane to any given accurary $\alpha$. In part 2 we claim that if a suitable relationship between the parameters $B$ and $C$ is satisfied then a bound on the number of corrections can be proven in the general (nonseparable) case. The bound of part 2 is in terms of the margin-based quantity $\mathcal{D}_\gamma(\boldsymbol{u}; (\boldsymbol{x}, y)) = \max\{0, \gamma - \frac{y\,\boldsymbol{u}\cdot\boldsymbol{x}}{\|\boldsymbol{x}\|_p}\}$, $\gamma > 0$. (Here a $p$-indexing for $\mathcal{D}_\gamma$ is understood). $\mathcal{D}_\gamma$ is called *deviation* in [4] and *linear hinge loss* in [7].

Notice that $B$ and $C$ in part 1 do not meet the requirements given in part 2. On the other hand, in the separable case $B$ and $C$ chosen in part 2 do not yield a hyperplane which is arbitrarily (up to a small $\alpha$) close to the maximal margin hyperplane.

**Theorem 1** *Let* $\mathcal{W} = \{\boldsymbol{w} \in \mathcal{R}^n : \|\boldsymbol{w}\|_q \le 1\}$, $S = ((\boldsymbol{x}_1, y_1), ..., (\boldsymbol{x}_T, y_T)) \in (\mathcal{R}^n \times \{-1, +1\})^T$, *and* $\mathcal{M}$ *be the set of corrections of* $\text{ALMA}_p(\alpha; B, C)$ *running on* $S$ *(i.e., the set of trials* $t$ *such that* $\frac{y_t\,\boldsymbol{w}_k \cdot \boldsymbol{x}_t}{\|\boldsymbol{x}_t\|_p} \le (1-\alpha)\,\gamma_k$).

*1. Let* $\gamma^* = \max_{\boldsymbol{w} \in \mathcal{W}} \min_{t=1,...,T} \frac{y_t\,\boldsymbol{w}\cdot\boldsymbol{x}_t}{\|\boldsymbol{x}_t\|_p} > 0$. *Then* $\text{ALMA}_p(\alpha; \sqrt{8}/\alpha, \sqrt{2})$ *achieves the following bound[5] on* $|\mathcal{M}|$:

$$|\mathcal{M}| \le \frac{2\,(p-1)}{(\gamma^*)^2}\left(\frac{2}{\alpha} - 1\right)^2 + \frac{8}{\alpha} - 4 = O\left(\frac{p-1}{\alpha^2\,(\gamma^*)^2}\right). \tag{1}$$

*Furthermore, throughout the run of* $\text{ALMA}_p(\alpha; \sqrt{8}/\alpha, \sqrt{2})$ *we have* $\gamma_k \geq \gamma^*$. *Hence (1) is also an upper bound on the number of trials t such that* $\frac{y_t \, \boldsymbol{w}_k \cdot \boldsymbol{x}_t}{||\boldsymbol{x}_t||_p} \leq (1 - \alpha) \, \gamma^*$.

*2. Let the parameters B and C in Figure 1 satisfy the equation*[6]

$$C^2 + 2 (1 - \alpha) \, B \, C = 1.$$

*Then for any* $\boldsymbol{u} \in \mathcal{W}$, $\text{ALMA}_p(\alpha; B, C)$ *achieves the following bound on* $|\mathcal{M}|$, *holding for any* $\gamma > 0$, *where* $\rho^2 = \frac{p-1}{C^2 \gamma^2}$:

$$|\mathcal{M}| \leq \frac{1}{\gamma} \sum_{t \in \mathcal{M}} D_\gamma(\boldsymbol{u}; (\boldsymbol{x}_t, y_t)) + \frac{\rho^2}{2} + \sqrt{\frac{\rho^4}{4} + \frac{\rho^2}{\gamma} \sum_{t \in \mathcal{M}} D_\gamma(\boldsymbol{u}; (\boldsymbol{x}_t, y_t)) + \rho^2}.$$

*Observe that when* $\alpha = 1$ *the above inequality turns to a bound on the number of* mistaken *trials. In such a case the value of* $\gamma_k$ *(in particular, the value of B) is immaterial, while C is forced to be 1.* $\square$

When $p = 2$ the computations performed by $\text{ALMA}_p$ essentially involve only dot products (recall that $p = 2$ yields $q = 2$ and $\mathbf{f} = \mathbf{f}^{-1} = $ identity). Thus the generalization of $\text{ALMA}_2$ to the kernel case is quite standard. In fact, the linear combination $\boldsymbol{w}_{k+1} \cdot \boldsymbol{x}$ can be computed recursively, since $\boldsymbol{w}_{k+1} \cdot \boldsymbol{x} = \frac{\boldsymbol{w}_k \cdot \boldsymbol{x} + \eta_k \, y_t \boldsymbol{x}_t \cdot \boldsymbol{x}}{N_{k+1}}$. Here the denominator $N_{k+1}$ equals $\max\{1, ||\boldsymbol{w}'_k||_2\}$ and the norm $||\boldsymbol{w}'_k||_2$ is again computed recursively by $||\boldsymbol{w}'_k||_2^2 = ||\boldsymbol{w}'_{k-1}||_2^2 / N_k^2 + 2 \, \eta_k \, y_t \boldsymbol{w}_k \cdot \boldsymbol{x}_t + \eta_k^2 \, ||\boldsymbol{x}_t||_2^2$, where the dot product $\boldsymbol{w}_k \cdot \boldsymbol{x}_t$ is taken from the $k$-th correction (the trial where the $k$-th weight update did occur).

## 4   Experimental results

We did some experiments running $\text{ALMA}_2$ on the well-known MNIST OCR database.[7] Each example in this database consists of a 28×28 matrix representing a digitalized image of a handwritten digit, along with a $\{0,1,...,9\}$-valued label. Each entry in this matrix is a value in $\{0,1,...,255\}$, representing a grey level. The database has 60000 training examples and 10000 test examples. The best accuracy results for this dataset are those obtained by LeCun et al. [11] through boosting on top of the neural net LeNet4. They reported a test error rate of 0.7%. A soft margin SVM achieved an error rate of 1.1% [3].

In our experiments we used $\text{ALMA}_2(\alpha; \frac{1}{\alpha}, \sqrt{2})$ with different values of $\alpha$. In the following $\text{ALMA}_2(\alpha)$ is shorthand for $\text{ALMA}_2(\alpha; \frac{1}{\alpha}, \sqrt{2})$. We compared to SVMs, the Perceptron algorithm and the Perceptron-like algorithm ROMMA [12]. We followed closely the experimental setting described in [3, 4, 12]. We used a polynomial kernel $K$ of the form $K(\boldsymbol{x}, \boldsymbol{y}) = (1 + \boldsymbol{x} \cdot \boldsymbol{y})^d$, with $d = 4$. (This choice was best in [4] and was also made in [3, 12].) However, we did not investigate any careful tuning of scaling factors. In particular, we did not determine the best instance scaling factor $s$ for our algorithm (this corresponds to using the kernel $K(\boldsymbol{x}, \boldsymbol{y}) = (1 + \boldsymbol{x} \cdot \boldsymbol{y}/s)^d$). In our experiments we set $s = 255$. This was actually the best choice in [12] for the Perceptron algorithm. We reduced the 10-class problem to 10 binary problems. Classification is made according to the maximum output of the 10 binary classifiers. The results are summarized in Table 1. As in [4], the output of a binary classifier is based on either the *last* hypothesis produced by the algorithms (denoted by "last" in Table 1) or Helmbold and Warmuth's [9] leave-one-out *voted* hypothesis (denoted by "voted"). We refer the reader to [4] for details. We trained the algorithms by cycling up to 3 times ("epochs") over the training set. All the results shown in Table 1 are averaged over 10 random permutations of the training sequence. The columns marked

"Corr's" give the total number of corrections made in the training phase for the 10 labels. The first three rows of Table 1 are taken from [4, 12, 13]. The first two rows refer to the Perceptron algorithm,[8] while the third one refers to the best [9] noise-controlled (NC) version of ROMMA, called "aggressive ROMMA". Our own experimental results are given in the last six rows.

Among these Perceptron-like algorithms, $\text{ALMA}_2$ "voted" seems to be the most accurate. The standard deviations about our averages are reasonably small. Those concerning test errors range in (0.03%, 0.09%). These results also show how accuracy and running time (as well as sparsity) can be traded-off against each other in a transparent way. The accuracy of our algorithm is slightly worse than SVMs'. On the other hand, our algorithm is quite faster and easier to implement than previous implementations of SVMs, such as those given in [17, 5]. An interesting features of $\text{ALMA}_2$ is that its approximate solution relies on fewer support vectors than the SVM solution.

We found the accuracy of 1.77 for $\text{ALMA}_2(1.0)$ fairly remarkable, considering that it has been obtained by sweeping through the examples just once for each of the ten classes. In fact, the algorithm is rather fast: training for one epoch the ten binary classifiers of $\text{ALMA}_2(1.0)$ takes on average 2.3 hours and the corresponding testing time is on average about 40 minutes. (All our experiments have been performed on a PC with a single Pentium® III MMX processor running at 447 Mhz.)

## 5  Concluding Remarks

In the full paper we will give more extensive experimental results for $\text{ALMA}_2$ and $\text{ALMA}_p$ with $p > 2$. One drawback of $\text{ALMA}_p$'s approximate solution is the absence of a bias term (i.e., a nonzero threshold). This seems to make little difference for MNIST dataset, but there are cases when a biased maximal margin hyperplane generalizes quite better than an unbiased one. It is not clear to us how to incorporate the SVMs' bias term in our algorithm. We leave this as an open problem.

Table 1: Experimental results on MNIST database. "TestErr" denotes the fraction of misclassified patterns in the test set, while "Corr's" gives the total number of training corrections for the 10 labels. Recall that voting takes place during the testing phase. Thus the number of corrections of "last" is the same as the number of corrections of "voted".

| | | 1 Epoch | | 2 Epochs | | 3 Epochs | |
|---|---|---|---|---|---|---|---|
| | | TestErr | Corr's | TestErr | Corr's | TestErr | Corr's |
| Perceptron | "last" | 2.71% | 7901 | 2.14% | 10421 | 2.03% | 11787 |
| | "voted" | 2.23% | 7901 | 1.86% | 10421 | 1.76% | 11787 |
| agg-ROMMA(NC) ("last") | | 2.05% | 30088 | 1.76% | 44495 | 1.67% | 58583 |
| $\text{ALMA}_2(1.0)$ | "last" | 2.52% | 7454 | 2.01% | 9658 | 1.86% | 10934 |
| | "voted" | 1.77% | 7454 | 1.52% | 9658 | 1.47% | 10934 |
| $\text{ALMA}_2(0.9)$ | "last" | 2.10% | 9911 | 1.74% | 12711 | 1.64% | 14244 |
| | "voted" | 1.69% | 9911 | 1.49% | 12711 | 1.40% | 14244 |
| $\text{ALMA}_2(0.8)$ | "last" | 1.98% | 12810 | 1.72% | 16464 | 1.60% | 18528 |
| | "voted" | 1.68% | 12810 | 1.44% | 16464 | 1.35% | 18528 |

**Acknowledgments**

Thanks to Nicolò Cesa-Bianchi, Nigel Duffy, Dave Helmbold, Adam Kowalczyk, Yi Li, Nick Littlestone and Dale Schuurmans for valuable conversations and email exchange. We would also like to thank the NIPS2000 anonymous reviewers for their useful comments and suggestions. The author is supported by a post-doctoral fellowship from Università degli Studi di Milano.

## Footnotes

[1]In fact, algorithms such as ROMMA and the one contained in Kowalczyk [10] have been specifically designed for euclidean norm. Any straightforward extension of these algorithms to a general norm $p$ seems to require numerical methods.

[2]We assume that $w \cdot x = 0$ yields a wrong classification, independent of $y$.

[3]The $p$-norm Perceptron algorithm is a generalization of the classical Perceptron algorithm [18]: $p$-norm Perceptron is actually Perceptron when $p = 2$.

[4]In the degenerate case that $\boldsymbol{x}_t = \boldsymbol{0}$ no update takes place.

[5]We did not optimize the constants here.

[6]Notice that $B$ and $C$ in part 1 do not satisfy this relationship.

[7]Available on Y. LeCun's home page: http://www.research.att.com/~yann/ocr/mnist/.

[8]These results have been obtained with no noise control. It is not clear to us how to incorporate any noise control mechanism into the classical Perceptron algorithm. The method employed in [10, 12] does not seem helpful in this case, at least for the first epoch.

[9]According to [12], ROMMA's last hypothesis seems to perform better than ROMMA's voted hypothesis.

# References

[1]   M. Anthony, P. Bartlett, *Neural Network Learning: Theoretical Foundations*, CMU, 1999.

[2]   P. Auer and C. Gentile Adaptive and self-confident on-line learning algorithms. In *13th COLT*, 107–117, 2000.

[3]   C. Cortes, V. Vapnik. Support-vector networks. *Machine Learning*, 20, 3: 273–297, 1995.

[4]   Y. Freund and R. Schapire. Large margin classification using the perceptron algorithm. *Journal of Machine Learning*, 37, 3: 277–296, 1999.

[5]   T.-T. Friess, N. Cristianini, and C. Campbell. The kernel adatron algorithm: a fast and simple learning procedure for support vector machines. In *15th ICML*, 1998.

[6]   C. Gentile and N. Littlestone. The robustness of the $p$-norm algorithms. In *12th COLT*, 1–11, 1999.

[7]   C. Gentile, and M. K. Warmuth. Linear hinge loss and average margin. In *11th NIPS*, 225–231, 1999.

[8]   A. J. Grove, N. Littlestone, and D. Schuurmans. General convergence results for linear discriminant updates. In *10th COLT*, 171–183, 1997.

[9]   D. Helmbold and M. K. Warmuth. On weak learning. *JCSS*, 50, 3: 551–573, 1995.

[10]  A. Kowalczyk. Maximal margin perceptron. In Smola, Bartlett, Scholkopf, and Schuurmans editors, Advances in large margin classifiers, MIT Press, 1999.

[11]  Y. Le Cun, L.J. Jackel, L. Bottou, A. Brunot, C. Cortes, J.S. Denker, H. Drucker, I. Guyon, U. Muller, S. Sackinger, P. Simard, and V. Vapnik, Comparison of learning algorithms for handwritten digit recognition. In *ICANN*, 53–60, 1995.

[12]  Y. Li, and P. Long. The relaxed online maximum margin algorithm. In *12th NIPS*, 498–504, 2000.

[13]  Y. Li. From support vector machines to large margin classifiers, PhD Thesis, School of Computing, the National University of Singapore, 2000.

[14]  N. Littlestone. Learning quickly when irrelevant attributes abound: A new linear-threshold algorithm. *Machine Learning*, 2:285–318, 1988.

[15]  O. Mangasarian, Mathematical programming in data mining. *Data Mining and Knowledge Discovery*, 42, 1: 183–201, 1997.

[16]  P. Nachbar, J.A. Nossek, J. Strobl, The generalized adatron algorithm. In *Proc. 1993 IEEE ISCAS*, 2152-5, 1993.

[17]  J. C. Platt. Fast training of support vector machines using sequential minimal optimization. In Scholkopf, Burges and Smola editors, Advances in kernel methods: support vector machines, MIT Press, 1998.

[18]  F. Rosenblatt. *Principles of neurodynamics: Perceptrons and the theory of brain mechanisms.* Spartan Books, Washington, D.C., 1962.
